# Hybrid NN/HMM-Based Speech Recognition with a Discriminant Neural Feature Extraction

**Daniel Willett, Gerhard Rigoll**

Department of Computer Science
Faculty of Electrical Engineering
Gerhard-Mercator-University Duisburg, Germany
{willett,rigoll}@fb9-ti.uni-duisburg.de

## Abstract

In this paper, we present a novel hybrid architecture for continuous speech recognition systems. It consists of a continuous HMM system extended by an arbitrary neural network that is used as a preprocessor that takes several frames of the feature vector as input to produce more discriminative feature vectors with respect to the underlying HMM system. This hybrid system is an extension of a state-of-the-art continuous HMM system, and in fact, it is the first hybrid system that really is capable of outperforming these standard systems with respect to the recognition accuracy. Experimental results show an relative error reduction of about 10% that we achieved on a remarkably good recognition system based on continuous HMMs for the Resource Management 1000-word continuous speech recognition task.

## 1 INTRODUCTION

Standard state-of-the-art speech recognition systems utilize Hidden Markov Models (HMMs) to model the acoustic behavior of basic speech units like phones or words. Most commonly the probabilistic distribution functions are modeled as mixtures of Gaussian distributions. These mixture distributions can be regarded as output nodes of a Radial-Basis-Function (RBF) network that is embedded in the HMM system [1]. Contrary to neural training procedures the parameters of the HMM system, including the RBF network, are usually estimated to maximize the training observations' likelihood. In order to combine the time-warping abilities of HMMs and the more discriminative power of neural networks, several hybrid approaches arose during the past five years, that combine HMM systems and neural networks. The best known approach is the one proposed by Bourlard [2]. It replaces the HMMs' RBF-net with a Multi-Layer-Perceptron (MLP) which is trained to output each HMM state's posterior probability. At last year's NIPS our group presented a novel hybrid speech recognition approach that combines a discrete HMM speech recognition system and a neural quantizer [3]. By maximizing the mutual information between the VQ-labels and the assigned phoneme-classes, this approach outperforms standard discrete recognition systems. We showed that this approach is capable of building up very accurate systems with an extremely fast likelihood computation, that only consists of a quantization and a table lookup. This resulted in a hybrid system with recognition performance equivalent to the best

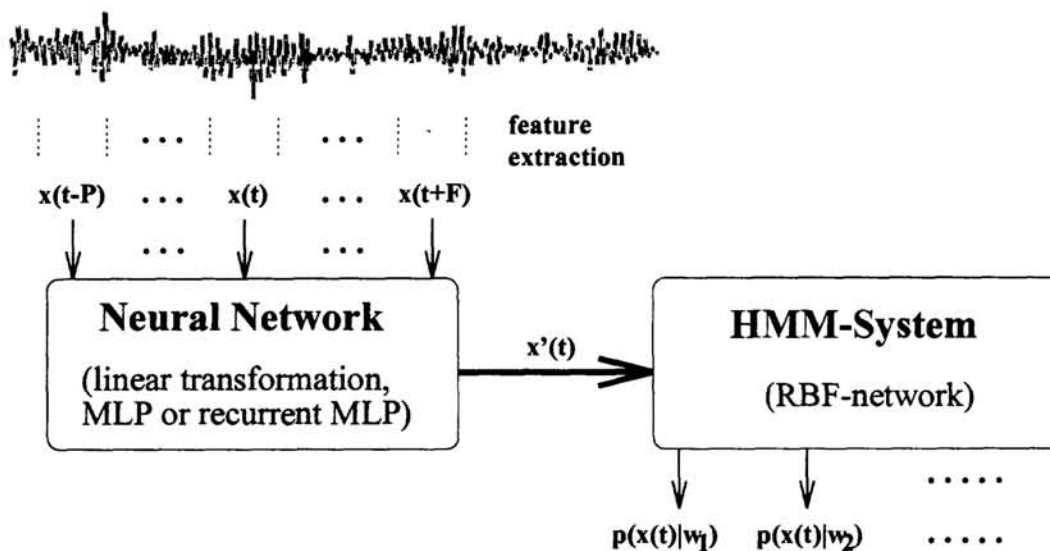

Figure 1: Architecture of the hybrid NN/HMM system

continuous systems, but with a much faster decoding. Nevertheless, it has turned out that this hybrid approach is not really capable of substantially outperforming very good continuous systems with respect to the recognition accuracy. This observation is similar to experiences with Bourlard's MLP approach. For the decoding procedure, this architecture offers a very efficient pruning technique (phone deactivation pruning [4]) that is much more efficient than pruning on likelihoods, but until today this approach did not outperform standard continuous HMM systems in recognition performance.

## 2   HYBRID CONTINUOUS HMM/MLP APPROACH

Therefore, we followed a different approach, namely the extension of a state-of-the-art continuous system that achieves extremely good recognition rates with a neural net that is trained with MMI-methods related to those in [5]. The major difference in this approach is the fact that the acoustic processor is not replaced by a neural network, but that the Gaussian probability density component is retained and combined with a neural component in an appropriate manner. A similar approach was presented in [6] to improve a speech recognition system for the TIMIT database. We propose to regard the additional neural component as being part of the feature extraction, and to reuse it in recognition systems of higher complexity where discriminative training is extremely expensive.

### 2.1   ARCHITECTURE

The basic architecture of this hybrid system is illustrated in Figure 1. The neural net functions as a feature transformation that takes several additional past and future feature vectors into account to produce an improved more discriminant feature vector that is fed into the HMM system. This architecture allows (at least) three ways of interpretation; 1. as a hybrid system that combines neural nets and continuous HMMs, 2. as an LDA-like transformation that incorporates the HMM parameters into the calculation of the transformation matrix and 3. as feature extraction method, that allows the extraction of features according to the underlying HMM system. The considered types of neural networks are linear transformations, MLPs and recurrent MLPs. A detailed description of the possible topologies is given in Section 3.
With this architecture, additional past and future feature vectors can be taken into account in the probability estimation process without increasing the dimensionality of the Gaussian mixture components. Instead of increasing the HMM system's number of parameters the neural net is trained to produce more discriminant feature vectors with respect to the trained HMM system. Of course, adding some kind of neural net increases the number of parameters too, but the increase is much more moderate than it would be when increasing each Gaussian's dimensionality.

## 2.2 TRAINING OBJECTIVE

The original purpose of this approach was the intention to transfer the hybrid approach presented in [3], based on MMI neural network, to (semi-) continuous systems. This way, we hoped to be able to achieve the same remarkable improvements that we obtained on discrete systems now on continuous systems, which are the much better and more flexible baseline systems. The most natural way to do this would be the re-estimation of the codebook of Gaussian mean vectors of a semi-continuous system using the neural MMI training algorithm presented in [3]. Unfortunately though, this won't work, as this codebook of a semi-continuous system does not determine a separation of the feature space, but is used as means of Gaussian densities. The MMI-principle can be retained, however, by leaving the original HMM system unmodified and instead extending it with a neural component, trained according to a frame-based MMI approach, related to the one in [3]. The MMI criterion is usually formulated in the following way:

$$\hat{\lambda}_{MMI} = \underset{\lambda}{\operatorname{argmax}} I_\lambda(X, W) = \underset{\lambda}{\operatorname{argmax}}(H_\lambda(X) - H_\lambda(X|W)) = \underset{\lambda}{\operatorname{argmax}} \frac{p_\lambda(X|W)}{p_\lambda(X)} \tag{1}$$

This means that following the MMI criterion the system's free parameters $\lambda$ have to be estimated to maximize the quotient of the observation's likelihood $p_\lambda(X|W)$ for the known transcription $W$ and its overall likelihood $p_\lambda(X)$. With $X = (x(1), x(2), ...x(T))$ denoting the training observations and $W = (w(1), w(2), ...w(T))$ denoting the HMM states - assigned to the observation vectors in a Viterbi-alignment - the frame-based MMI criterion becomes

$$\hat{\lambda}_{MMI} \approx \underset{\lambda}{\operatorname{argmax}} \sum_{i=1}^{T} I_\lambda(x(i), w(i))$$

$$= \underset{\lambda}{\operatorname{argmax}} \prod_{i=1}^{T} \frac{p_\lambda(x(i)|w(i))}{p_\lambda(x(i))} \approx \underset{\lambda}{\operatorname{argmax}} \prod_{i=1}^{T} \frac{p_\lambda(x(i)|w(i))}{\sum_{k=1}^{S} p_\lambda(x(i)|w_k)p(w_k)} \tag{2}$$

where $S$ is the total number of HMM states, $(w_1, ...w_S)$ denotes the HMM states and $p(w_k)$ denotes each states' prior-probability that is estimated on the alignment of the training data or by an analysis of the language model.

Eq. 2 can be used to re-estimate the Gaussians of a continuous HMM system directly. In [7] we reported the slight improvements in recognition accuracy that we achieved with this parameter estimation. However, it turned out, that only the incorporation of additional features in the probability calculation pipeline can provide more discriminative emission probabilities and a major advance in recognition accuracy. Thus, we experienced it to be more convenient to train an additional neural net in order to maximize Eq. 2. Besides, this approach offers the possibility of improving a recognition system by applying a trained feature extraction network taken from a different system. Section 5 will report our positive experiences with this procedure.

At first, for matter of simplicity, we will consider a linear network that takes P past feature vectors and F future feature vectors as additional input. With the linear net denoted as a $(P + F + 1) \times N$ matrix $NET$, each component $x'(t)[c]$ of the network output $x'(t)$ computes to

$$x'(t)[c] = \sum_{i=0}^{P+F} \sum_{j=1}^{N} x(t - P + i)[j] \cdot NET[i * N + j][c] \qquad \forall c \in \{1...N\} \tag{3}$$

so that the derivative with respect to a component of $NET$ easily computes to

$$\frac{\partial x'(t)[c]}{\partial NET[i * N + j][\hat{c}]} = \delta_{c,\hat{c}} x(t - P + i)[j] \tag{4}$$

In a continuous HMM system with diagonal covariance matrices the pdf of each HMM state $w$ is modeled by a mixture of Gaussian components like

$$p_\lambda(x|w) = \sum_{j=1}^{C_w} d_{wj} \frac{1}{\sqrt{(2\pi)^n |\sigma_j|}} e^{-\frac{1}{2} \sum_{l=1}^{N} \frac{(m_j[l]-x[l])^2}{\sigma_j[l]}} \tag{5}$$

A pdf's derivative with respect to a component $x'[c]$ of the net's output becomes

$$\frac{\partial p_\lambda(x'|w)}{\partial x'[c]} = \sum_{j=1}^{C_w} d_{wj} \frac{(x[c] - m_j[c])}{\sigma_j[c]} \frac{1}{\sqrt{(2\pi)^n |\sigma_j|}} e^{-\frac{1}{2} \sum_{l=1}^{N} \frac{(m_j[l] - x'[l])^2}{\sigma_j[l]}} \qquad (6)$$

With $x(t)$ in Eq. 2 now replaced by the net output $x'(t)$ the partial derivative of Eq. 2 with respect to a probabilistic distribution function $p(x'(i)|w_k)$ computes to

$$\frac{\partial I_\lambda(x'(i), w(i))}{\partial p_\lambda(x'(i)|w_k)} = \frac{\delta_{w(i),w_k}}{p_\lambda(x(i)|w_k)} - \frac{p(w_k)}{\sum_{l=1}^{S} p_\lambda(x(i)|w_l) p(w_l)} \qquad (7)$$

Thus, using the chain rule the derivative of the net's parameters with respect to the frame-based MMI criterion can be computed as displayed in Eq. 8

$$\frac{\partial I_\lambda(X, W)}{\partial NET[l][c]} = \sum_{i=1}^{T} \left( \sum_{k=1}^{S} \left( \frac{\partial I_\lambda(x(i)|w(i)))}{\partial p_\lambda(x'(i)|w_k)} \frac{\partial p_\lambda(x'(i)|w_k)}{\partial x'(i)[c]} \frac{\partial x'(i)[c]}{\partial NET[l][c]} \right) \right) \qquad (8)$$

and a gradient descent procedure can be used to determine the optimal parameter estimates.

## 2.3 ADVANTAGES OF THE PROPOSED APPROACH

When using a linear network, the proposed approach strongly resembles the well known Linear Discriminant Analysis (LDA) [8] in architecture and training objective. The main difference is the way the transformation is set up. In the proposed approach the transformation is computed by taking directly the HMM parameters into account whereas the LDA only tries to separate the features according to some class assignment. With the incorporation of a trained continuous HMM system the net's parameters are estimated to produce feature vectors that not only have a good separability in general, but also have a distribution that can be modeled with mixtures of Gaussians very well. Our experiments given at the end of this paper prove this advantage. Furthermore, contrary to LDA, that produces feature vectors that don't have much in common with the original vectors, the proposed approach only slightly modifies the input vectors. Thus, a well trained continuous system can be extended by the MMI-net approach, in order to improve its recognition performance without the need for completely rebuilding it. In addition to that, the approach offers a fairly easy extension to nonlinear networks (MLP) and recurrent networks (recurrent MLP). This will be outlined in the following Section. And, maybe as the major advantage, the approach allows keeping up the division of the input features into streams of features that are strongly uncorrelated and which are modeled with separate pdfs. The case of multiple streams is discussed in detail in Section 4. Besides, the MMI approach offers the possibility of a unified training of the HMM system and the feature extraction network or an iterative procedure of training each part alternately.

## 3 NETWORK TOPOLOGIES

Section 2 explained how to train a linear transformation with respect to the frame-based MMI criterion. However, to exploit all the advantages of the proposed hybrid approach the network should be able to perform a nonlinear mapping, in order to produce features whose distribution is (closer to) a mixture of Gaussians although the original distribution is not.

### 3.1 MLP

When using a fully connected MLP as displayed in Figure 2 with one hidden layer of $H$ nodes, that perform the nonlinear function $f$, the activation of one of the output nodes $x'(t)[c]$ becomes

$$x'(t)[c] = \sum_{h=1}^{H} L2[h][c] \cdot f\left( BIAS_h + \sum_{i=0}^{P+F} \sum_{j=1}^{N} x(t - P + i)[j] \cdot L1[i * N + j][h] \right) \qquad (9)$$

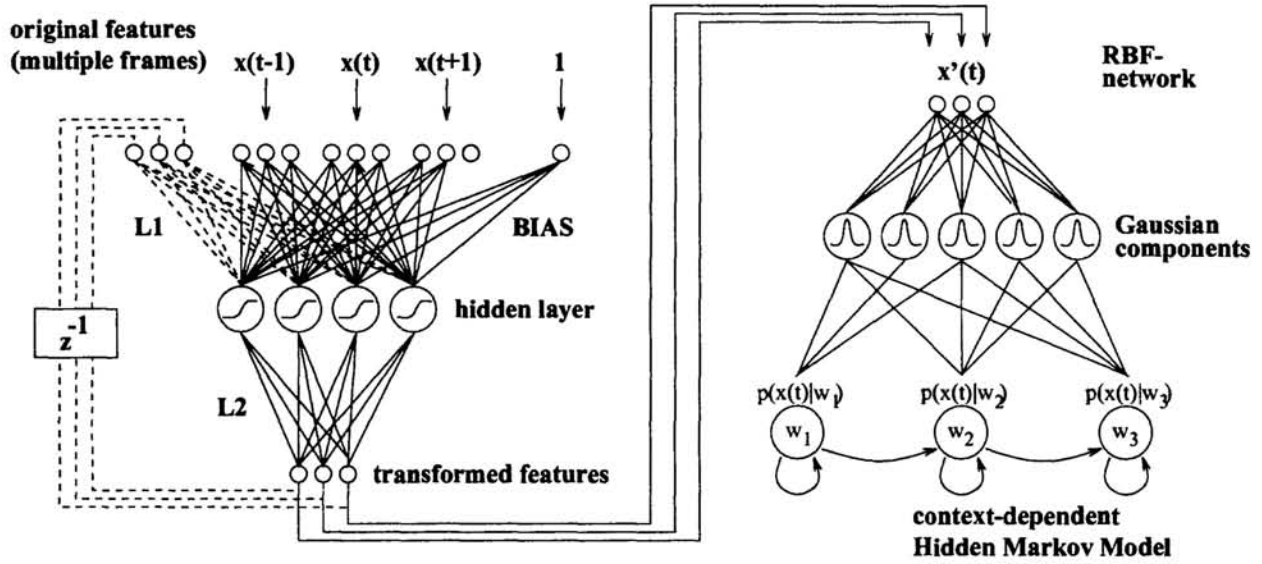

Figure 2: Hybrid system with a nonlinear feature transformation

which is easily differentiable with respect to the nonlinear network's parameters. In our experiments we chose $f$ to be defined as the hyperbolic tangents $f(x) := tanh(x) = (2(1 + e^{-x})^{-1} - 1)$ so that the partial derivative with respect to i.e. a weight $L1[\hat{i} \cdot N + \hat{j}][h]$ of the first layer computes to

$$\frac{\partial x'(t)[c]}{\partial L1[\hat{i} \cdot N + \hat{j}][h]} = x(t - P + \hat{i})[\hat{j}] \cdot L2[h][c] \tag{10}$$

$$\cdot cosh\left(BIAS_h + \sum_{i=0}^{P+F} \sum_{j=1}^{N} x(t - P + i)[j] \cdot L1[i * N + j][h]\right)^{-2}$$

and the gradient can be set up according to Eq. 8.

## 3.2 RECURRENT MLP

With the incorporation of several additional past feature vectors as explained in Section 2, more discriminant feature vectors can be generated. However, this method is not capable of modeling longer term relations, as it can be achieved by extending the network with some recurrent connections. For the sake of simplicity, in our experiments we simply extended the MLP as indicated with the dashed lines in Figure 2 by propagating the output $x(t)$ back to the input of the network (with a delay of one discrete time step). This type of recurrent neural net is often referred to as a 'Jordan'-network. Certainly, the extension of the network with additional hidden nodes in order to model the recurrence more independently would be possible as well.

## 4 MULTI STREAM SYSTEMS

In HMM-based recognition systems the extracted features are often divided into streams that are modeled independently. This is useful the less correlated the divided features are. In this case the overall likelihood of an observation computes to

$$p_\lambda(x|w) = \prod_{s=1}^{M} p_{s\lambda}(x|w)^{w_s} \tag{11}$$

where each of the stream pdfs $p_{s\lambda}(x|w)$ only uses a subset of the features in $x$. The stream weights $w_s$ are usually set to unity.

| system | baseline system | LDA | linear MMI-net | MLP (H=36) | Jordan-Network |
|---|---|---|---|---|---|
| monophones one stream | 24% | 21% | 21% | 21% | - |
| monophones four streams | 11.8% | 11.0% | 10.9% | 10.8% | 10,9% |
| triphones four streams | 5.2% | 5.3% | 4.8% | 4.7% | 4.7% |

Table 1: Word error rates achieved in the experiments

A multi stream system can be improved by a neural extraction for each stream and an independent training of these neural networks. However, it has to be considered that the subdivided features usually are not totally independent and by considering multiple input frames as illustrated in Figure 1 this dependence often increases. It is a common practice, for instance, to model the features' first and second order delta coefficients in independent streams. So, for sure the streams lose independence when considering multiple frames, as these coefficients are calculated using the additional frames. Nevertheless, we found it to give best results to maintain this subdivision into streams, but to consider the stronger correlation by training each stream's net dependent on the other nets' outputs. A training criterion follows straight from Eq. 11 inserted in Eq. 2.

$$\hat{\lambda}_{MMI} = \operatorname*{argmax}_{\lambda} \prod_{i=1}^{T} \frac{p_{\lambda}(x(i)|w(i))}{p_{\lambda}(x(i))} = \operatorname*{argmax}_{\lambda} \prod_{i=1}^{T} \prod_{s=1}^{M} \left( \frac{p_{s\lambda}(x(i)|w(i))}{p_{s\lambda}(x(i))} \right)^{w_s} \quad (12)$$

The derivative of this equation with respect to the pdf $p_{\hat{s}\lambda}(x|w)$ of a specific stream $\hat{s}$ depends on the other streams' pdfs. With the $w_s$ set to unity it is

$$\frac{\partial I_{\lambda}(x'(i), w(i))}{\partial p_{\hat{s}\lambda}(x'(i)|w_k)} = \left( \prod_{s \neq \hat{s}} \frac{p_{s\lambda}(x(i)|w(i))}{p_{s\lambda}(x(i))} \right) \left( \frac{\delta_{w(i),w_k}}{p_{\hat{s}\lambda}(x(i)|w_k)} - \frac{p(w_k)}{\sum_{l=1}^{S} p_{\hat{s}\lambda}(x(i)|w_l)p(w_l)} \right)$$

$$(13)$$

Neglecting the correlation among the streams the training of each stream's net can be done independently. However, the more the incorporation of additional features increases the streams' correlation, the more important it gets to train the nets in a unified training procedure according to Eq. 13.

## 5  EXPERIMENTS AND RESULTS

We applied the proposed approach to improve a context-independent (monophones) and a context-dependent (triphones) continuous speech recognition system for the 1000-word Resource Management (RM) task. The systems used linear HMMs of three emitting states each. The tying of Gaussian mixture components was performed with an adaptive procedure according to [9]. The HMM states of the word-internal triphone system were clustered in a tree-based phonetic clustering procedure. Decoding was performed with a Viterbi-decoder and the standard wordpair-grammar of perplexity 60. Training of the MLP was performed with the RPROP algorithm. For training the weights of the recurrent connections we chose real-time recurrent learning. The average error rates were computed using the test-sets Feb89, Oct89, Feb91 and Sep92.

The table above shows the recognition results with single stream systems in its first section. These systems simply use a 12-value Cepstrum feature vector without the incorporation of delta coefficients. The systems with an input transformation use one additional past and one additional future feature vector as input. The proposed approach achieves the same performance as the LDA, but it is not capable of outperforming it.

The second section of the table lists the recognition results with four stream systems that use the first and second order delta coefficients in additional streams plus log energy and this values' delta coefficients in a forth stream. The MLP system trained according to Eq.

11 slightly outperforms the other approaches. The incorporation of recurrent network connections does not improve the system's performance.

The third section of the table lists the recognition results with four stream systems with a context-dependent acoustic modeling (triphones). The applied LDA and the MMI-networks were taken from the monophone four stream system. On the one hand, this was done to avoid the computational complexity that the MMI training objective causes on context-dependent systems. On the other hand, this demonstrates that the feature vectors produced by the trained networks have a good discrimination for continuous systems in general. Again, the MLP system outperforms the other approaches and achieves a very remarkable word error rate. It should be pointed out here, that the structure of the continuous system as reported in [9] is already highly optimized and it is almost impossible to further reduce the error rate by means of any acoustic modeling method. This is reflected in the fact that even a standard LDA cannot improve this system. Only the new neural approach leads to a 10% reduction in error rate which is a large improvement considering the fact that the error rate of the baseline system is among the best ever reported for the RM database,

## 6 CONCLUSION

The paper has presented a novel approach to discriminant feature extraction. A MLP network has successfully been used to compute a feature transformation that outputs extremely suitable features for continuous HMM systems. The experimental results have proven that the proposed approach is an appropriate method for including several feature frames in the probability estimation process without increasing the dimensionality of the Gaussian mixture components in the HMM system. Furthermore did the results on the triphone speech recognition system prove that the approach provides discriminant features, not only for the system that the mapping is computed on, but for HMM systems with a continuous modeling in general. The application of recurrent networks did not improve the recognition accuracy. The longer range relations seem to be very weak and they seem to be covered well by using the neighboring feature vectors and first and second order delta coefficients. The proposed unified training procedure for multiple nets in multi-stream systems allows keeping up the subdivision of features of weak correlations, and gave us best profits in recognition accuracy.

## References

[1] H. Ney, "Speech Recognition in a Neural Network Framework: Discriminative Training of Gaussian Models and Mixture Densities as Radial Basis Functions", *Proc. IEEE-ICASSP,* 1991, pp. 573–576.

[2] H Bourlard, N. Morgan, "Connectionist Speech Recognition - A Hybrid Approach", *Kluwer Academic Press,* 1994.

[3] G. Rigoll, C. Neukirchen, "A new approach to hybrid HMM/ANN speech recognition using mutual information neural networks", *Advances in Neural Information Processing Systems (NIPS-96),* Denver, Dec. 1996, pp. 772–778.

[4] M. M. Hochberg, G. D. Cook, S. J. Renals, A. J. Robinson, A. S. Schechtman, "The 1994 ABBOT Hybrid Connectionist-HMM Large-Vocabulary Recognition System", *Proc. ARPA Spoken Language Systems Technology Workshop,* 1995.

[5] G. Rigoll, "Maximum Mutual Information Neural Networks for Hybrid Connectionist-HMM Speech Recognition", *IEEE-Trans. Speech Audio Processing,* Vol. 2, No. 1, Jan. 1994, pp. 175–184.

[6] Y. Bengio et al., "Global Optimization of a Neural Network - Hidden Markov Model Hybrid" *IEEE-Transcations on NN,* Vol. 3, No. 2, 1992, pp. 252–259.

[7] D. Willett, C. Neukirchen, R. Rottland, "Dictionary-Based Discriminative HMM Parameter Estimation for Continuous Speech Recognition Systems", *Proc. IEEE-ICASSP,* 1997, pp. 1515–1518.

[8] X. Aubert, R. Haeb-Umbach, H. Ney, "Continuous mixture densities and linear discriminant analysis for improved context-dependent acoustic models", *Proc. IEEE-ICASSP,* 1993, pp. II 648–651.

[9] D. Willett, G. Rigoll, "A New Approach to Generalized Mixture Tying for Continuous HMM-Based Speech Recognition", *Proc. EUROSPEECH,* Rhodes, 1997.